# Neural Control for Nonlinear Dynamic Systems

**Ssu-Hsin Yu**
Department of Mechanical Engineering
Massachusetts Institute of Technology
Cambridge, MA 02139
Email: hsin@mit.edu

**Anuradha M. Annaswamy**
Department of Mechanical Engineering
Massachusetts Institute of Technology
Cambridge, MA 02139
Email: aanna@mit.edu

## Abstract

A neural network based approach is presented for controlling two distinct types of nonlinear systems. The first corresponds to nonlinear systems with parametric uncertainties where the parameters occur nonlinearly. The second corresponds to systems for which stabilizing control structures cannot be determined. The proposed neural controllers are shown to result in closed-loop system stability under certain conditions.

## 1 INTRODUCTION

The problem that we address here is the control of general nonlinear dynamic systems in the presence of uncertainties. Suppose the nonlinear dynamic system is described as $\dot{x} = f(x, u, \theta)$, $y = h(x, u, \theta)$ where $u$ denotes an external input, $y$ is the output, $x$ is the state, and $\theta$ is the parameter which represents constant quantities in the system. The control objectives are to stabilize the system in the presence of disturbances and to ensure that reference trajectories can be tracked accurately, with minimum delay. While uncertainties can be classified in many different ways, we focus here on two scenarios. One occurs because the changes in the environment and operating conditions introduce uncertainties in the system parameter $\theta$. As a result, control objectives such as regulation and tracking, which may be realizable using a continuous function $u = \gamma(x, \theta)$ cannot be achieved since $\theta$ is unknown. Another class of problems arises due to the complexity of the nonlinear function $f$. Even if $\theta$, $f$ and $h$ can be precisely determined, the selection of an appropriate $\gamma$ that leads to stabilization and tracking cannot be made in general. In this paper, we present two methods based on neural networks which are shown to be applicable to both the above classes of problems. In both cases, we clearly outline the assumptions made, the requirements for adequate training of the neural network, and the class of engineering problems where the proposed methods are applicable. The proposed approach significantly expands the scope of neural controllers in relation to those proposed in (Narendra and Parthasarathy, 1990; Levin and Narendra, 1993; Sanner and Slotine, 1992; Jordan and Rumelhart, 1992).

The first class of problems we shall consider includes nonlinear systems with parametric uncertainties. The field of adaptive control has addressed such a problem, and over the past thirty years, many results have been derived pertaining to the control of both linear and nonlinear dynamic systems (Narendra and Annaswamy, 1989). A common assumption in almost all of the published work in this field is that the uncertain parameters occur linearly. In this paper, we consider the control of nonlinear dynamic systems with nonlinear parametrizations. We design a neural network based controller that adapts to the parameter $\theta$ and show that closed-loop system stability can be achieved under certain conditions. Such a controller will be referred to as a $\theta$-adaptive neural controller. Pertinent results to this class are discussed in section 2.

The second class of problems includes nonlinear systems, which despite being completely known, cannot be stabilized by conventional analytical techniques. The obvious method for stabilizing nonlinear systems is to resort to linearization and use linear control design methods. This limits the scope of operation of the stabilizing controller. Feedback linearization is another method by which nonlinear systems can be stably controlled (Isidori, 1989). This however requires fairly stringent set of conditions to be satisfied by the functions $f$ and $h$. Even after these conditions are satisfied, one cannot always find a closed-form solution to stabilize the system since it is equivalent to solving a set of partial differential equations. We consider in this paper, nonlinear systems, where system models as well as parameters are known, but controller structures are unknown. A neural network based controller is shown to exist and trained so that a stable closed-loop system is achieved. We denote this class of controllers as a stable neural controller. Pertinent results to this class are discussed in section 3.

## 2  $\theta$-ADAPTIVE NEURAL CONTROLLER

The focus of the nonlinear adaptive controller to be developed in this paper is on dynamic systems that can be written in the $d$-step ahead predictor form as follows:

$$y_{t+d} = f_r(\omega_t, u_t, \theta) \tag{1}$$

where $\omega_t^T = [y_t, \cdots, y_{t-n+1}, u_{t-1}, \cdots, u_{t-m-d+1}]$, $n \geq 1$, $m \geq 0$, $d \geq 1$, $m + d = n$, $\mathcal{Y}_1, \mathcal{U}_1 \subset \Re$ containing the origin and $\Theta_1 \subset \Re^k$ are open, $f_r : \mathcal{Y}_1^n \times \mathcal{U}_1^{m+d} \times \Theta_1 \to \Re$, $y_t$ and $u_t$ are the output and the input of the system at time $t$ respectively, and $\theta$ is an unknown parameter and occurs nonlinearly in $f_r$.[1] The goal is to choose a control input $u$ such that the system in (1) is stabilized and the plant output is regulated around zero.

Let $x_t^T \triangleq [y_{t+d-1}, \cdots, y_{t+1}, \omega_t^T]^T$, $A_m = [e_2, \cdots, e_{n+d-1}, 0, e_{n+d+1}, \cdots, e_{n+m+2d-2}, 0]$, $B_m = [e_1, e_{n+d}]$, where $e_i$ is an unit vector with the $i$-th term equal to 1. The following assumptions are made regarding the system in Eq. (1).

(A1) For every $\theta \in \Theta_1$, $f_r(0, 0, \theta) = 0$.

(A2) There exist open and convex neighborhoods of the origin $\mathcal{Y}_2 \subset \mathcal{Y}_1$ and $\mathcal{U}_2 \subset \mathcal{U}_1$, an open and convex set $\Theta_2 \subset \Theta_1$, and a function $K : \Omega_2 \times \mathcal{Y}_2 \times \Theta_2 \to \mathcal{U}_1$ such that for every $\omega_t \in \Omega_2$, $y_{t+d} \in \mathcal{Y}_2$ and $\theta \in \Theta_2$, Eq. (1) can be written as $u_t = K(\omega_t, y_{t+d}, \theta)$, where $\Omega_2 \triangleq \mathcal{Y}_2^n \times \mathcal{U}_2^{m+d-1}$.

(A3) $K$ is twice differentiable and has bounded first and second derivatives on $E_1 \triangleq \Omega_2 \times \mathcal{Y}_2 \times \Theta_2$, while $f_r$ is differentiable and has a bounded derivative on $\Omega_2 \times K(E_1) \times \Theta_2$.

(A4) There exists $\delta_g > 0$ such that for every $y_1 \in f_r(\Omega_2, K(\Omega_2, 0, \Theta_2), \Theta_2)$, $\omega \in \Omega_2$ and
$\theta, \widehat{\theta} \in \Theta_2$, $|1 - (\frac{\partial K(\omega, y, \theta)}{\partial y} - \frac{\partial K(\omega, y, \widehat{\theta})}{\partial y})|_{y=y_1} \cdot \frac{\partial f_r(\omega, u, \theta)}{\partial u}|_{u=u_1}| > \delta_g$.

(A5) There exist positive definite matrices $P$ and $Q$ of dimensions $(n + m + 2d - 2)$ such that $x_t^T(A_m^T P A_m - P)x_t + \bar{K}^T B_m^T P B_m \bar{K} + 2x_t^T A_m^T P B_m \bar{K} \leq -x_t^T Q x_t$, where $\bar{K} = [0, K(\omega_t, 0, \theta)]^T$.

Since the objective is to control the system in (1) where $\theta$ is unknown, in order to stabilize the output $y$ at the origin with an estimate $\widehat{\theta}_t$, we choose the control input as

$$u_t = K(\omega_t, 0, \widehat{\theta}_t) \tag{2}$$

## 2.1 PARAMETER ESTIMATION SCHEME

Suppose the estimation algorithm for updating $\widehat{\theta}_t$ is defined recursively as $\Delta\widehat{\theta}_t \triangleq \widehat{\theta}_t - \widehat{\theta}_{t-1} = R(y_t, \omega_{t-d}, u_{t-d}, \widehat{\theta}_{t-1})$ the problem is to determine the function $R$ such that $\widehat{\theta}_t$ converges to $\theta$ asymptotically. In general, $R$ is chosen to depend on $y_t$, $\omega_{t-d}$, $u_{t-d}$ and $\widehat{\theta}_{t-1}$ since they are measurable and contain information regarding $\theta$. For example, in the case of linear systems which can be cast in the input predictor form, $u_t = \phi_t^T \theta$, a well-known linear parameter estimation method is to adjust $\Delta\widehat{\theta}$ as (Goodwin and Sin, 1984) $\Delta\widehat{\theta}_t = \frac{\phi_{t-d}}{1+\phi_{t-d}^T\phi_{t-d}}[u_{t-d} - \phi_{t-d}^T\widehat{\theta}_{t-1}]$. In other words, the mechanism for carrying out parameter estimation is realized by $R$. In the case of general nonlinear systems, the task of determining such a function $R$ is quite difficult, especially when the parameters occur nonlinearly. Hence, we propose the use of a neural network parameter estimation algorithm denoted $\theta$-adaptive neural network (TANN) (Annaswamy and Yu, 1996). That is, we adjust $\widehat{\theta}_t$ as

$$\widehat{\theta}_t = \begin{cases} \widehat{\theta}_{t-1} + N(y_t, \omega_{t-d}, u_{t-d}, \widehat{\theta}_{t-1}) & \text{if } \Delta V_{d_t} < -\epsilon \\ \widehat{\theta}_{t-1} & \text{otherwise} \end{cases} \tag{3}$$

where the inputs of the neural network are $y_t$, $\omega_{t-d}$, $u_{t-d}$ and $\widehat{\theta}_{t-1}$, the output is $\Delta\widehat{\theta}_t$, and $\epsilon$ defines a dead-zone where parameter adaptation stops.

The neural network is to be trained so that the resulting network can improve the parameter estimation over time for any possible $\theta$ in a compact set. In addition, the trained network must ensure that the overall system in Eqs. (1), (2) and (3) is stable. Toward this end, $N$ in TANN algorithm is required to satisfy the following two properties: (P1) $|N(y_t, \omega_{t-d}, u_{t-d}, \widehat{\theta}_{t-1})|^2 \leq a\frac{|C(\bar{\phi}_{t-d})|^2}{(1+|C(\bar{\phi}_{t-d})|^2)^2}\tilde{u}_{t-d}^2$, and (P2) $\Delta V_t - \Delta V_{d_t} < \epsilon_1$, $\epsilon_1 > 0$, where $\Delta V_t = |\tilde{\theta}_t|^2 - |\tilde{\theta}_{t-1}|^2$, $\tilde{\theta}_t = \widehat{\theta}_t - \theta$, $\Delta V_{d_t} = -a\frac{2+|C(\bar{\phi}_{t-d})|^2}{(1+|C(\bar{\phi}_{t-d})|^2)^2}\tilde{u}_{t-d}^2$, $C(\bar{\phi}_t) = \left(\frac{\partial K}{\partial\theta}(\omega_t, y_{t+d}, \theta)\big|_{\theta=\theta_0}\right)^T$, $\tilde{u}_t = u_t - K(\omega_t, y_{t+d}, \widehat{\theta}_{t+d-1})$, $\bar{\phi}_t = [\omega_t^T, y_{t+d}]^T$, $a \in (0, 1)$ and $\theta_0$ is the point where $K$ is linearized and often chosen to be the mean value of parameter variation.

## 2.2 TRAINING OF TANN FOR CONTROL

In the previous section, we proposed an algorithm using a neural network for adjusting the control parameters. We introduced two properties (P1) and (P2) of the identification algorithm that the neural network needs to possess in order to maintain stability of the closed-loop system. In this section, we discuss the training procedure by which the weights of the neural network are to be adjusted so that the network retains these properties.

The training set is constructed off-line and should compose of data needed in the training phase. If we want the algorithm in Eq. (3) to be valid on the specified sets $\mathcal{Y}_3$ and $\mathcal{U}_3$ for various $\theta$ and $\widehat{\theta}$ in $\Theta_3$, the training set should cover those variables appearing in Eq. (3) in their respective ranges. Hence, we first sample $\omega$ in the set $\mathcal{Y}_3^n \times \mathcal{U}_3^{m+d-1}$,

and $\theta$, $\widehat{\theta}$ in the set $\Theta_3$. Their values are, say, $\omega_1$, $\theta_1$ and $\widehat{\theta}_1$ respectively. For the particular $\widehat{\theta}_1$ and $\theta_1$ we sample $\widehat{\theta}$ again in the set $\{\theta \in \Theta_3| \ |\theta - \theta_1| \leq |\widehat{\theta}_1 - \theta_1|\}$, and its value is, say, $\widehat{\theta}_1^d$. Once $\omega_1$, $\theta_1$, $\widehat{\theta}_1$ and $\widehat{\theta}_1^d$ are sampled, other data can then be calculated, such as $u_1 = K(\omega_1, 0, \widehat{\theta}_1)$ and $y_1 = f_r(\omega_1, u_1, \theta_1)$. We can also obtain the corresponding $C(\bar{\phi}_1) = \frac{\partial K}{\partial \theta}(\omega_1, y_1, \theta_0)$, $\Delta V_{d_1} = -a\frac{2+|C(\bar{\phi}_1)|^2}{(1+|C(\bar{\phi}_1)|^2)^2}(u_1 - \widehat{u}_1)^2$ and $L_1 = a\frac{|C(\bar{\phi}_1)|^2}{(1+|C(\bar{\phi}_1)|^2)^2}(u_1 - \widehat{u}_1)^2$, where $\bar{\phi}_1 = [\omega_1^T, y_1]^T$ and $\widehat{u}_1 = K(\omega_1, y_1, \widehat{\theta}_1^d)$. A data element can then be formed as $(y_1, \omega_1, u_1, \widehat{\theta}_1^d, \theta_1, \Delta V_{d_1}, L_1)$. Proceeding in the same manner, by choosing various $\omega_s$, $\theta_s$, $\widehat{\theta}_s$ and $\widehat{\theta}_s^d$ in their respective ranges, we form a typical training set $T_{train} = \left\{(y_s, \omega_s, u_s, \widehat{\theta}_s^d, \theta_s, \Delta V_{d_s}, L_s)| \ 1 \leq s \leq M\right\}$, where $M$ denotes the total number of patterns in the training set. If the quadratic penalty function method (Bertsekas, 1995) is used, properties (P1) and (P2) can be satisfied by training the network on the training set to minimize the following cost function:

$$\min_W J \triangleq \min_W \frac{1}{2}\sum_{i=1}^{M}\left\{\left(\max\{0, \Delta V_{e_i}\}\right)^2 + \frac{1}{b^2}\left(\max\{0, |N_i(W)|^2 - L_i\}\right)^2\right\} \quad (4)$$

To find a $W$ which minimizes the above unconstrained cost function $J$, we can apply algorithms such as the gradient method and the Gauss-Newton method.

## 2.3  STABILITY RESULT

With the plant given by Eq. (1), the controller by Eq. (2), and the TANN parameter estimation algorithm by Eq. (3), it can be shown that the stability of the closed-loop system is guaranteed.

Based on the assumptions of the system in (1) and properties (P1) and (P2) that TANN satisfies, the stability result of the closed-loop system can be concluded in the following theorem. We refer the reader to (Yu and Annaswamy, 1996) for further detail.

**Theorem 1** *Given the compact sets $\mathcal{Y}_3^{n+1} \times \mathcal{U}_3^{m+d} \times \Theta_3$ where the neural network in Eq. (3) is trained. There exist $\epsilon_1, \epsilon > 0$ such that for any interior point $\theta$ of $\Theta_3$, there exist open sets $\mathcal{Y}_4 \subset \mathcal{Y}_3$, $\mathcal{U}_4 \subset \mathcal{U}_3$ and a neighborhood $\Theta_4$ of $\theta$ such that if $y_0, \cdots, y_{n+d-2} \in \mathcal{Y}_4$, $u_0, \cdots, u_{n-2} \in \mathcal{U}_4$, and $\widehat{\theta}_{n-1}, \cdots, \widehat{\theta}_{n+d-2} \in \Theta_4$, then all the signals in the closed-loop system remain bounded and $y_t$ converges to a neighborhood of the origin.*

## 2.4  SIMULATION RESULTS

In this section, we present a simulation example of the TANN controller proposed in this section. The system is of the form $y_{t+1} = \frac{\theta y_t(1-y_t)}{1+e^{-0.05\theta u_t}} + u_t$, where $\theta$ is the parameter to be determined on-line. Prior information regarding the system is that $\theta \in [4, 10]$. Based on Eq. (2), the controller was chosen to be $u_t = -\frac{\widehat{\theta}_t y_t(1-y_t)}{1+e^{-0.05\widehat{\theta}_t u_t}}$, where $\widehat{\theta}_t$ denotes the parameter estimate at time $t$. According to Eq. (3), $\theta$ was estimated using the TANN algorithm with inputs $y_{t+1}$, $y_t$, $u_t$ and $\widehat{\theta}_t$, and $\epsilon = 0.01$. $N$ is a Gaussian network with 700 centers. The training set and the testing set were composed of 6,040 and 720 data elements respectively.

After the training was completed, we tested the TANN controller on the system with six different values of $\theta$, 4.5, 5.5, 6.5, 7.5, 8.5 and 9.5, while the initial parameter estimate and the initial output were chosen as $\widehat{\theta}_1 = 7$ and $y_0 = -0.9$ respectively. The results are plotted in Figure 1. It can be seen that $y_t$ can be stabilized at the origin for all these values of $\theta$. For comparison, we also simulated the system under the same conditions but with $\widehat{\theta}$

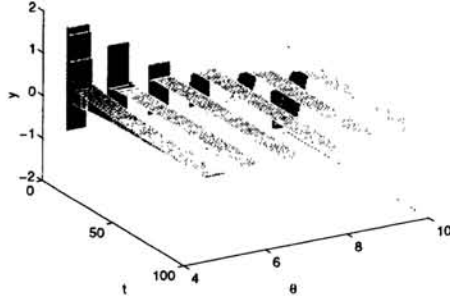

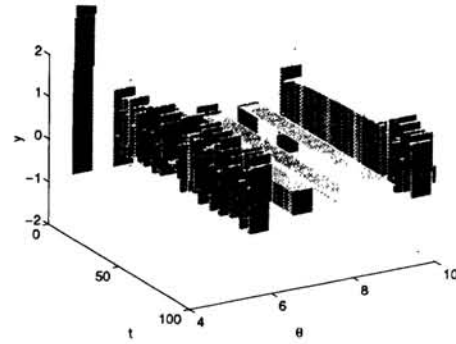

Figure 1: $y_t$ (TANN Controller)                    Figure 2: $y_t$ (Extended Kalman Filter)

estimated using the extended Kalman filter (Goodwin and Sin, 1984). Figure 2 shows the output responses. It is not surprising that for some values of $\theta$, especially when the initial estimation error is large, the responses either diverge or exhibit steady state error.

# 3  STABLE NEURAL CONTROLLER

## 3.1  STATEMENT OF THE PROBLEM

Consider the following nonlinear dynamical system

$$\dot{x} = f(x, u), \qquad y = h(x) \tag{5}$$

where $x \in R^n$ and $u \in R^m$. Our goal is to construct a stabilizing neural controller as $u = N(y; W)$ where $N$ is a neural network with weights $W$, and establish the conditions under which the closed-loop system is stable.

The nonlinear system in (5) is expressed as a combination of a higher-order linear part and a nonlinear part as $\dot{x} = Ax + Bu + R_1(x, u)$ and $y = Cx + R_2(x)$, where $f(0, 0) = 0$ and $h(0) = 0$. We make the following assumptions: (A1) $f, h$ are twice continuously differentiable and are completely known. (A2) There exists a $K$ such that $(A - BKC)$ is asymptotically stable.

## 3.2  TRAINING OF THE STABLE NEURAL CONTROLLER

In order for the neural controller in Section 3.1 to result in an asymptotically stable closed-loop system, it is sufficient to establish that a continuous positive definite function of the state variables decreases monotonically through output feedback. In other words, if we can find a scalar definite function with a negative definite derivative of all points in the state space, we can guarantee stability of the overall system. Here, we limit our choices of the Lyapunov function candidates to the quadratic form, i.e. $V = x^T P x$, where $P$ is positive definite, and the goal is to choose the controller so that $\dot{V} < 0$ where $\dot{V} = 2x^T P f(x, N(h(x), W))$.

Based on the above idea, we define a "desired" time-derivative $\dot{V}_d$ as $\dot{V}_d = -x^T Q x$ where $Q = Q^T > 0$. We choose $P$ and $Q$ matrices as follows. First, according to (A1), we can find a matrix $K$ to make $(A - BKC)$ asymptotically stable. We can then find a $(P, Q)$ pair by choosing an arbitrary positive definite matrix $Q$ and solving the Lyapunov equation, $(A - BKC)^T P + P(A - BKC) = -Q$ to obtain a positive definite $P$.

With the controller of the form in Section 3.1, the goal is to find $W$ in the neural network which yields $\dot{V} \leq \dot{V}_d$ along the trajectories in a neighborhood $\mathcal{X} \subset \Re^n$ of the origin in the state space. Let $x_i$ denote the value of a sample point where $i$ is an index to the sample variable $x \in \mathcal{X}$ in the state space. To establish $\dot{V} \leq \dot{V}_d$, it is necessary that for every $x_i$ in a neighborhood $\mathcal{X} \subset \Re^n$ of the origin, $\dot{V}_i \leq \dot{V}_{d_i}$, where $\dot{V}_i = 2x_i^T P f(x_i, N(h(x_i), W))$ and $\dot{V}_{d_i} = -x_i^T Q x_i$. That is, the goal is to find a $W$ such that the inequality constraints $\Delta V_{e_i} \leq 0$, where $i = 1, \cdots, M$, is satisfied, where $\Delta V_{e_i} = \dot{V}_i - \dot{V}_{d_i}$ and $M$ denotes the total number of sample points in $\mathcal{X}$. As in the training of TANN controller, this can also be posed as an optimization problem. If the same quadratic penalty function method is used, the problem is to find $W$ to minimize the following cost function over the training set, which is described as $T_{train} = \{(x_i, y_i, \dot{V}_{d_i}) | 1 \leq i \leq M\}$:

$$\min_W J \triangleq \min_W \frac{1}{2} \sum_{i=1}^{M} \left( \max\{0, \Delta V_{e_i}\} \right)^2 \tag{6}$$

## 3.3  STABILITY OF THE CLOSED-LOOP SYSTEM

Assumptions (A1) and (A2) imply that a stabilizing controller $u = -Ky$ exists so that $V = x^T P x$ is a candidate Lyapunov function. More generally, suppose a continuous but unknown function $\gamma(y)$ exists such that for $V = x^T P x$, a control input $u = \gamma(y)$ leads to $\dot{V} \leq -x^T Q x$, then we can find a neural network $N(y)$ which approximates $\gamma(y)$ arbitrarily closely in a compact set leading to closed-loop stability. This is summarized in Theorem 2 (Yu and Annaswamy, 1995).

**Theorem 2** *Let there be a continuous function $\gamma(h(x))$ such that $2x^T P f(x, \gamma(h(x))) + x^T Q x \leq 0$ for every $x \in \mathcal{X}$ where $\mathcal{X}$ is a compact set containing the origin as an interior point. Then, given a neighborhood $\mathcal{O} \subset \mathcal{X}$ of the origin, there exists a neural controller $u = N(h(x); W)$ and a compact set $\mathcal{Y} \in \mathcal{X}$ such that the solutions of $\dot{x} = f(x, N(h(x); W))$ converge to $\mathcal{O}$, for every initial condition $x(t_0) \in \mathcal{Y}$.*

## 3.4  SIMULATION RESULTS

In this section, we show simulation results for a discrete-time nonlinear systems using the proposed neural network controller in Section 3, and compare it with a linear controller to illustrate the difference. The system we considered is a second-order nonlinear system $x_t = f(x_{t-1}, u_{t-1})$, where $f = [f_1, f_2]^T$, $f_1 = x_{1_{t-1}} \times (1 + x_{2_{t-1}}) + x_{2_{t-1}} \times (1 - u_{t-1} + u_{t-1}^2)$ and $f_2 = x_{1_{t-1}}^2 + 2x_{2_{t-1}} + u_{t-1}(1 + x_{2_{t-1}})$. It was assumed that $x$ is measurable, and we wished to stabilize the system around the origin. The controller is of the form $u_t = N(x_{1_t}, x_{2_t})$. The neural network $N$ used is a Gaussian network with 120 centers. The training set and the testing set were composed of 441 and 121 data elements respectively.

After the training was done, we plotted the actual change of the Lyapunov function, $\Delta V$, using the linear controller $u = -Kx$ and the neural network controller in Figures 3 and 4 respectively. It can be observed from the two figures that if the neural network controller is used, $\Delta V$ is negative definite except in a small neighborhood of the origin, which assures that the closed-loop system would converge to vicinity of the origin; whereas, if the linear controller is used, $\Delta V$ becomes positive in some region away from the origin, which implies that the system can be unstable for some initial conditions. Simulation results confirmed our observation.

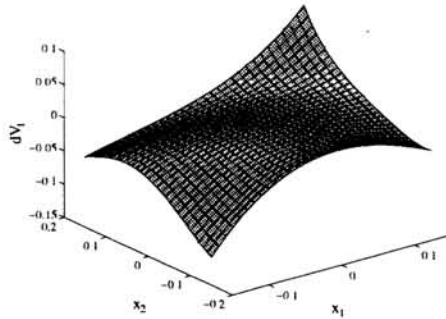
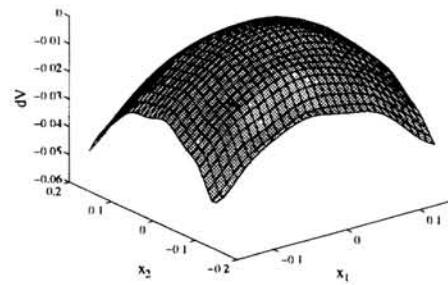

Figure 3: $\Delta V(u = -Kx)$               Figure 4: $\Delta V(u = N(x))$

**Acknowledgments**

This work is supported in part by Electrical Power Research Institute under contract No. 8060-13 and in part by National Science Foundation under grant No. ECS-9296070.

## Footnotes

[1]Here, as well as in the following sections, $A^n$ denotes the $n$-th product space of the set $A$.

## References

[1] A. M. Annaswamy and S. Yu. $\theta$-adaptive neural networks: A new approach to parameter estimation. *IEEE Transactions on Neural Networks*, (to appear) 1996.

[2] D. P. Bertsekas. *Nonlinear Programming*. Athena Scientific, Belmont, MA, 1995.

[3] G. C. Goodwin and K. S. Sin. *Adaptive Filtering Prediction and Control*. Prentice-Hall, Inc., 1984.

[4] A. Isidori. *Nonlinear Control Systems*. Springer-Verlag, New York, NY, 1989.

[5] M. I. Jordan and D. E. Rumelhart. Forward models: Supervised learning with a distal teacher. *Cognitive Science*, 16:307–354, 1992.

[6] A. U. Levin and K. S. Narendra. Control of nonlinear dynamical systems using neural networks: Controllability and stabilization. *IEEE Transactions on Neural Networks*, 4(2):192–206, March 1993.

[7] K. S. Narendra and A. M. Annaswamy. *Stable Adaptive Systems*. Prentice-Hall, Inc., 1989.

[8] K. S. Narendra and K. Parthasarathy. Identification and control of dynamical systems using neural networks. *IEEE Transactions on Neural Networks*, 1(1):4–26, March 1990.

[9] R. M. Sanner and J.-J. E. Slotine. Gaussian networks for direct adaptive control. *IEEE Transactions on Neural Networks*, 3(6):837–863, November 1992.

[10] S. Yu and A. M. Annaswamy. Adaptive control of nonlinear dynamic systems using $\theta$-adaptive neural networks. Technical Report 9601, Adaptive Control Laboratory, Department of Mechanical Engineering, M.I.T., 1996.

[11] S.-H. Yu and A. M. Annaswamy. Control of nonlinear dynamic systems using a stability based neural network approach. In *Technical report 9501, Adaptive Control Laboratory, MIT, Submitted to Proceedings of the 34th IEEE Conference on Decision and Control*, New Orleans, LA, 1995.